# Application of Blind Separation of Sources to Optical Recording of Brain Activity

**Holger Schöner, Martin Stetter, Ingo Schießl**
Department of Computer Science
Technical University of Berlin Germany
{*hfsch,moatl,ingos*}@*cs.tu-berlin.de*

**John E.W. Mayhew**
University of Sheffield, UK
*j.e.mayhew@sheffield.ac.uk*

**Jennifer S. Lund, Niall McLoughlin**
Institute of Ophthalmology
University College London, UK
{*j.lund,n.mcloughlin*}@*ucl.ac.uk*

**Klaus Obermayer**
Department of Computer Science,
Technical University of Berlin, Germany
*oby@cs.tu-berlin.de*

## Abstract

In the analysis of data recorded by optical imaging from intrinsic signals (measurement of changes of light reflectance from cortical tissue) the removal of noise and artifacts such as blood vessel patterns is a serious problem. Often bandpass filtering is used, but the underlying assumption that a spatial frequency exists, which separates the mapping component from other components (especially the global signal), is questionable. Here we propose alternative ways of processing optical imaging data, using blind source separation techniques based on the spatial decorrelation of the data. We first perform benchmarks on artificial data in order to select the way of processing, which is most robust with respect to sensor noise. We then apply it to recordings of optical imaging experiments from macaque primary visual cortex. We show that our BSS technique is able to extract ocular dominance and orientation preference maps from single condition stacks, for data, where standard post-processing procedures fail. Artifacts, especially blood vessel patterns, can often be completely removed from the maps. In summary, our method for blind source separation using extended spatial decorrelation is a superior technique for the analysis of optical recording data.

## 1 Introduction

One approach in the attempt of comprehending how the human brain works is the analysis of neural activation patterns in the brain for different stimuli presented to a sensory system. An example is the extraction of ocular dominance or orientation preference maps from recordings of activity of neurons in the primary visual cortex of mammals. A common technique for extracting such maps is optical imaging (OI) of intrinsic signals. Currently this is the imaging technique with the highest spatial resolution ($\approx 100\,\mu$m) for mapping of the cortex. This method is explained e.g. in [1], for similar methods using voltage sensitive dyes see [2, 3]. OI uses changes in light reflection to estimate spatial patterns of stimulus

answers. The overall change recorded by a CCD or video camera is the total signal. The part of the total signal due to local neural activity is called the mapping component and it derives from changes in deoxyhemoglobin absorption and light scattering properties of the tissue. Another component of the total signal is a "global" component, which is also correlated with stimulus presentation, but has a much coarser spatial resolution. It derives its part from changes in the blood volume with the time. Other components are blood vessel artifacts, the vasomotor signal (slow oscillations of neural activity), and ongoing activity (spontaneous, stimulus-uncorrelated activity). Problematic for the extraction of activity maps are especially blood vessel artifacts and sensor noise, such as photon shot noise. A procedure often used for extracting the activity maps from the recordings is bandpass filtering, after preprocessing by temporal, spatial, and trial averaging. Lowpass filtering is unproblematic, as the spatial resolution of the mapping signal is limited by the scattering properties of the brain tissue, hence everything above a limiting frequency must be noise. The motivation for highpass filtering, on the other hand, is questionable as there is no specific spatial frequency separating local neural activity patterns and the global signal [4]).

A different approach, Blind Source Separation (BSS), models the components of the recorded image frames as independent sources, and the observations (recorded image frames) as noisy linear mixtures of the unknown sources. After performing the BSS the mapping component should ideally be concentrated in one estimated source, the global signal in another, and blood vessel artifacts, etc. in further ones. Previous work ([5]) has shown that BSS algorithms, which are based on higher order statistics ([6, 7, 8]), fail for optical imaging data, because of the high signal to noise ratio.

In this work we suggest and investigate versions of the M&S algorithm [9, 10], which are robust against sensor noise, and we analyze their performance on artificial as well as real optical recording data. In section 2 we describe an improved algorithm, which we later compare to other methods in section 3. There an artificial data set is used for the analysis of noise robustness, and benchmark results are presented. Then, in section 4, it is shown that the newly developed algorithm is very well able to separate the different components of the optical imaging data, for ocular dominance as well as orientation preference data from monkey striate cortex. Finally, section 5 provides conclusions and perspectives for future work.

## 2    Second order blind source separation

Let $m$ be the number of mixtures and $\mathbf{r}$ the sample index, i.e. a vector specifying a pixel in the recorded images. The observation vectors $\mathbf{y}(\mathbf{r}) = (y_1(\mathbf{r}), \dots, y_m(\mathbf{r}))^T$ are assumed to be linear mixtures of $m$ unknown sources $\mathbf{s}(\mathbf{r}) = (s_1(\mathbf{r}), \dots, s_m(\mathbf{r}))^T$ with A being the $m \times m$ mixing matrix and $\mathbf{n}$ describing the sensor noise:

$$\mathbf{y}(\mathbf{r}) = \mathbf{A}\mathbf{s}(\mathbf{r}) + \mathbf{n} \tag{1}$$

The goal of BSS is to obtain optimal source estimates $\hat{\mathbf{s}}(\mathbf{r})$ under the assumption that the original sources are independent. In the noiseless case $\mathbf{W} = \mathbf{A}^{-1}$ would be the optimal demixing matrix. In the noisy case, however, $\mathbf{W}$ also has to compensate for the added noise: $\hat{\mathbf{s}}(\mathbf{r}) = \mathbf{W}\mathbf{y}(\mathbf{r}) = \mathbf{W} \cdot \mathbf{A} \cdot \mathbf{s}(\mathbf{r}) + \mathbf{W} \cdot \mathbf{n}$. BSS algorithms are generally only able to recover the original sources up to a permutation and scaling.

Extended Spatial Decorrelation (ESD) uses the second order statistics of the observations to find the source estimates. If sources are statistical independent all source cross-correlations

$$C_{i,j}^{(s)}(\Delta\mathbf{r}) = \langle s_i(\mathbf{r})s_j(\mathbf{r} + \Delta\mathbf{r})\rangle_{\mathbf{r}} = \frac{1}{R}\sum_{\mathbf{r}} s_i(\mathbf{r})s_j(\mathbf{r} + \Delta\mathbf{r}) \qquad , i \neq j \tag{2}$$

must vanish for all shifts $\Delta \mathbf{r}$, while the autocorrelations ($i = j$) of the sources remain (the variances). Note that this implies that the sources must be spatially smooth.

Motivated by [10] we propose to optimize the cost function, which is the sum of the squared cross-correlations of the estimated sources over a set of shifts $\{\Delta \mathbf{r}\}$,

$$E(\mathbf{W}) = \sum_{\Delta \mathbf{r}} \sum_{i \neq j} \left( \left( \mathbf{W} \mathbf{C}(\Delta \mathbf{r}) \mathbf{W}^T \right)_{i,j} \right)^2 \tag{3}$$

$$= \sum_{\Delta \mathbf{r}} \sum_{i \neq j} \langle \hat{s}_i(\mathbf{r}) \hat{s}_j(\mathbf{r} + \Delta \mathbf{r}) \rangle_{\mathbf{r}}^2,$$

with respect to the demixing matrix $\mathbf{W}$. The matrix $C_{i,j}(\Delta \mathbf{r}) = \langle y_i(\mathbf{r}) y_j(\mathbf{r} + \Delta \mathbf{r}) \rangle_{\mathbf{r}}$ denotes the mixture cross-correlations for a shift $\Delta \mathbf{r}$. This cost function is minimized using the Polak Ribiere Conjugate Gradient technique, where the line search is substituted by a dynamic step width adaptation ([11]). To keep the demixing matrix $\mathbf{W}$ from converging to the zero matrix, we introduce a constraint which keeps the diagonal elements of $\mathbf{T} = \mathbf{W}^{-1}$ (in the noiseless case and for non-sphered data $\mathbf{T}$ is an estimate of the mixing matrix, with possible permutations) at a value of $1.0$. Convergence properties are improved by sphering the data (transforming their correlation matrix for shift zero to an identity matrix) prior to decorrelating the mixtures.

Note that use of multiple shifts $\Delta \mathbf{r}$ allows to use more information about the auto- and cross-correlation structure of the mixtures for the separation process. Two shifts provide just enough constraints for a unique solution ([10]). Multiple shifts, and the redundancy they introduce, additionally allow to cancel out part of the noise by approximate simultaneous diagonalization of the corresponding cross correlation matrices.

In the presence of sensor noise, added after mixing, the standard sphering technique is problematic. When calculating the zero-shift cross-correlation matrix the variance of the noise contaminates the result, and sphering using a shifted cross-correlation matrix, is recommended ([12]). For spatially white sensor noise and sources with reasonable auto correlations this technique is more appropriate. In the following we denote the standard algorithm by dpa0, and the variant using noise robust sphering by dpa1.

## 3  Benchmarks for artificial data

The artificial data set used here, whose sources are approximately uncorrelated for all shifts, is shown in the left part of figure 1. The mixtures were produced by generating a random mixing matrix (in this case with condition number 3.73), applying it to the sources, and finally adding white noise of different variances.

In order to measure the performance on the artificial data set we measure a reconstruction error (RE) between the estimated and the correct sources via (see [13]):

$$\mathsf{RE}(\mathbf{W}) = \mathrm{od}\left( \sum_{\mathbf{r}} \hat{\mathbf{s}}(\mathbf{r}) \mathbf{s}^T(\mathbf{r}) \right), \quad \mathrm{od}(\mathbf{C}) = \frac{1}{N} \sum_i \frac{1}{N-1} \left( \sum_j \frac{|C_{i,j}|}{\max_k |C_{i,k}|} - 1 \right) \tag{4}$$

The correlation between the real and the estimated sources (the argument to "od"), should be close to a permutation matrix, if the separation is successful. If the maxima of two rows are in the same column, the separation is labeled unsuccessful. Otherwise, the normalized absolute sum of non-permutation (cross-correlation) elements is computed and returned as the reconstruction error.

We now compare the method based on optimization of (3) by gradient descent with the following variants of second order blind source separation: (1) standard spatial decorrelation

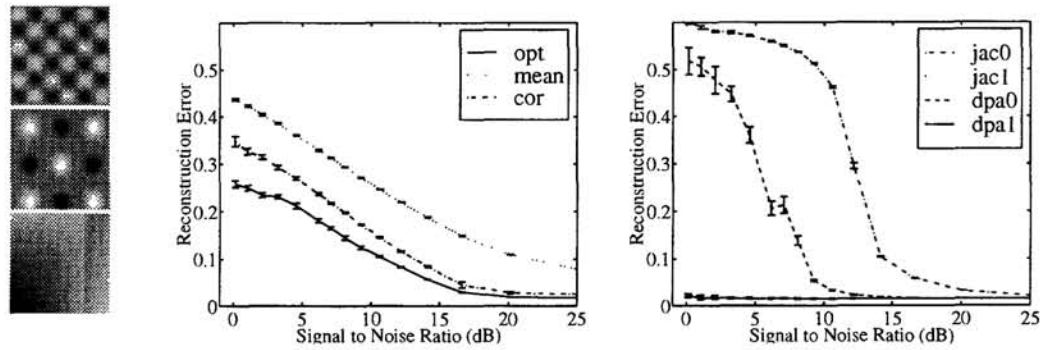

Figure 1: The set of three approximately uncorrelated source images of the artificial data set (left). The two plots (middle, right) show the reconstruction error versus signal to noise ratio for different separation algorithms. In the right plot jac1 and dpa1 are very close together.

using the optimal single shift yielding the smallest reconstruction error (opt). (2) Spatial decorrelation using the shift selected by

$$\Delta \mathbf{r}_{cor} = \text{argmax}_{\{\Delta \mathbf{r}\}} \frac{\text{norm}\left(\mathbf{C}(\Delta \mathbf{r}) - \text{diag}\left(\mathbf{C}(\Delta \mathbf{r})\right)\right)}{\text{norm}\left(\text{diag}\left(\mathbf{C}(\Delta \mathbf{r})\right)\right)} \cdot, \tag{5}$$

where "diag" sets all off-diagonal elements of its argument matrix to zero, and "norm" computes the largest singular value of its argument matrix (cor). $\Delta \mathbf{r}_{cor}$ is the shift for which the cross correlations are largest, i.e. whose signal to noise ratio (SNR) should be best. (3) Standard spatial decorrelation using the average reconstruction error for all successful shifts in a $61 \times 61$ square around the zero shift (mean). (4) A multi-shift algorithm ([12]), using several elementary rotations (Jacobi method) to build an orthogonal demixing matrix, which optimizes the cost function (3). The variants using standard sphering and noise robust sphering are denoted by (jac0) and (jac1). cor, opt, and mean use two shifts for their computation; but as one of those is always the zero-shift, there is only one shift to choose and they are called single-shift algorithms here.

Figure 1 gives two plots which show the reconstruction error (4) versus the SNR (measured in dB) for single shift (middle) and multi-shift (right) algorithms. The error bars indicate twice the standard error of the mean ($2\times$ SEM), for 10 runs with the same mixing matrix, but newly generated noise of the given noise level. In each of these runs, the best result of three was selected for the gradient descent method. This is because, contrary to the other algorithms, the gradient descent algorithm depends on the initial choice of the demixing matrix. All multi-shift algorithms (all except opt and mean), used 8 shifts $(\pm r, \pm r)$, $(\pm r, 0)$, and $(0, \pm r)$ for each $r \in \{1, 3, 5, 10, 20, 30\}$, so 48 all together.

Several points are noticeable in the plots. (i) The cor algorithm is generally closer to the optimum than to the average successful shift. (ii) A comparison between the two plots shows that the multi-shift algorithms (right plot) are able to perform much better than even the optimal single-shift method. For low to medium noise levels this is even the case when using the standard sphering method combined with the gradient descent algorithm. (iii) The advantage of the noise robust sphering method, compared to the standard sphering, is obvious: the reconstruction error stays very low for all evaluated noise levels, for both the jac1 and dpa1 algorithms. (iv) The gradient descent technique is more robust than the Jacobi method For the standard sphering its performance is much better than that of the Jacobi method.

Figure 1 shows results which were produced using a single mixing matrix. However, our simulations show that the algorithms compare qualitatively similar when using mixing ma-

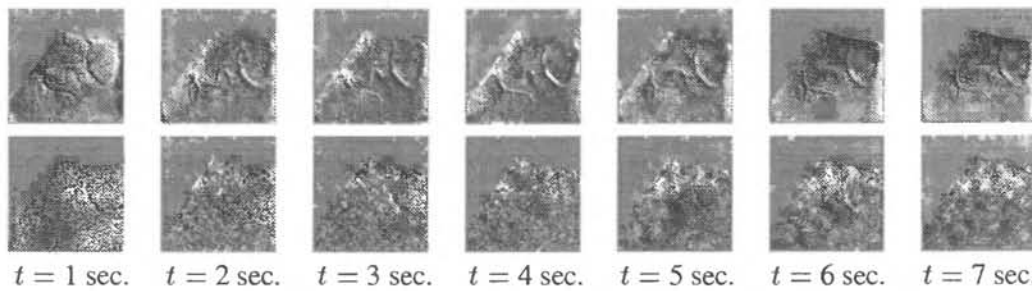

$t = 1$ sec.  $t = 2$ sec.  $t = 3$ sec.  $t = 4$ sec.  $t = 5$ sec.  $t = 6$ sec.  $t = 7$ sec.

Figure 2: Optical imaging stacks. The top stack is a single condition stack from ocular dominance experiments, the lower one a difference stack from orientation preference experiments (images for 90° gratings subtracted from those for 0° gratings). The stimulus was present during recording images 2-7 in each row. Two large blood vessels in the top and left regions of the raw images were masked out prior to the analysis.

trices with condition numbers between 2 and 10. The noise robust versions of the multi-shift algorithms generally yield the best separation results of all evaluated algorithms.

## 4  Application to optical imaging

We now apply extended spatial decorrelation to the analysis of optical imaging data. The data consists of recordings from the primary visual cortex of macaque monkeys. Each trial lasted 8 seconds, which were recorded with frame rates of 15 frames per second. A visual stimulus (a drifting bar grating of varying orientation) was presented between seconds 2 and 8. Trials were separated by a recovery period of 15 seconds without stimulation. The cortex was illuminated at a wavelength of 633 nm. One pixel corresponds to about 15 $\mu$m on the cortex; the image stacks used for further processing, consisting of $256 \times 256$ pixels, covered an area of cortex of approximately 3.7 mm$^2$.

Blocks of 15 consecutive frames were averaged, and averaging over 8 trials using the same visual stimulus further improved the SNR. First frame analysis (subtraction of the first, blank, frame from the others) was then applied to the resulting stack of 8 frames, followed by lowpass filtering with 14 cycles/mm. Figure 2 shows the resulting image stacks for an ocular dominance and an orientation preference experiment. One observes strong blood vessel artifacts (particularly in the top row of images), which are superimposed to the patchy mapping component that pops up over time.

Figure 3 shows results obtained by the application of extended spatial decorrelation (using dpa0). Only those estimated sources containing patterns different from white noise are shown. Backprojection of the estimated sources onto the original image stack yields the amplitude time series of the estimated sources, which is very useful in selecting the mapping component: it can be present in the recordings only after the stimulus onset (starting at $t = 2$ sec.). The middle part shows four estimated sources for the ocular dominance single condition stack. The mapping component (first image) is separated from the global component (second image) and blood vessel artifacts (second to fourth) quite well. The time course of the mapping component is plausible as well: calculation of a plausibility index (sum of squared differences between the normalized time series and a step function, which is 0 before and 1 after the stimulus onset) gives 0.5 for the mapping component and 2.31 for the next best one. Results for the gradient descent algorithm are similar for this data set, regardless of the sphering technique used. The Jacobi method also gives similar results, but a small blood vessel artifact is remaining in the resulting map. The cor algorithm usually gives much worse separation results. In the right part of figure 3 two es-

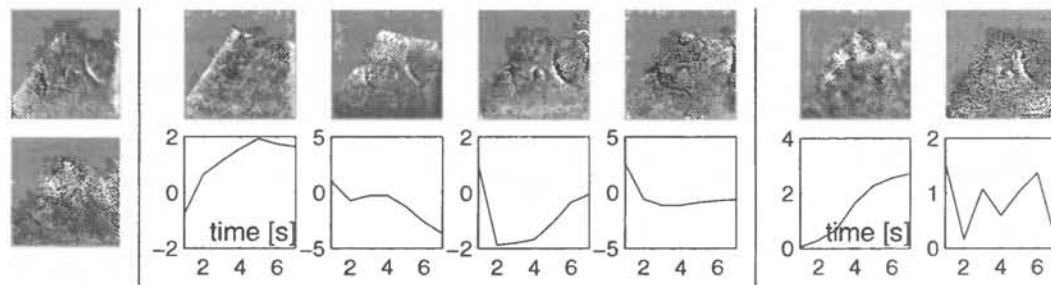

Figure 3: Left: Summation technique for ocular dominance (OD) experiment (upper) and orientation preference (OP) experiment (lower). Middle, Right: dpa0 algorithm applied to the same OD single condition (middle) and OP (right) stacks. The images show the 4 (OD) and 2 (OP) estimated components, which are visually different from white noise. In the bottom row the respective time courses of the estimated sources are given.

timated sources (those different from white noise) for the orientation preference difference stack can be seen. Here the proposed algorithm (dpa0) again works very well (plausibility index is 0.56 for mapping component, compared to 3.04 for the best other component). It generally has to be applied a few times (usually around 3 times) to select the best separation result (judging by visual quality of the separation and the time courses of the estimated sources), because of its dependence on parameter initialization; in return it yields the best results of all algorithms used, especially when compared to the traditional summation technique.

The similar results when using standard and noise robust sphering, and the small differences between the gradient descent and the Jacobi algorithms indicate, that not sensor noise is the limiting factor for the quality of the extracted maps. Instead it seems that, assuming a linear mixing model, no better results can be obtained from the used image stacks. It will remain for further research to analyze, how appropriate the linear mixing model is, and whether the underlying biophysical components are sufficiently uncorrelated. In the meantime the maps obtained by the ESD algorithm are superior to those obtained using conventional techniques like summation of the image stack.

## 5   Conclusion

The results presented in the previous sections show the advantages of the proposed algorithm: In the comparison with other spatial decorrelation algorithms the benefit in using multiple shifts compared to only two shifts is demonstrated. The robustness against sensor noise is improved, and in addition, the selection of multiple shifts is less critical than selecting a single shift, as the resulting multi-shift system of equations contains more redundancy. In comparison with the Jacobi method, which is restricted to find only orthogonal demixing matrices, the greater tolerance of demixing by a gradient descent technique concerning noise and incorrect sphering are demonstrated. The application of second order blind separation of sources to optical imaging data shows that these techniques represent an important alternative to the conventional approach, bandpass filtering followed by summation of the image stack, for extraction of neural activity maps. Vessel artifacts can be separated from the mapping component better than using classical approaches. The spatial decorrelation algorithms are very well adapted to the optical imaging task, because of their use of spatial smoothness properties of the mapping and other biophysical components.

An important field for future research concerning BSS algorithms is the incorporation of prior knowledge about sources and the mixing process, e.g. that the mixing has to be causal: the mapping signal cannot occur before the stimulus is presented. Assumptions

about the time course of signals could also be helpful, as well as knowledge about their spatial statistics. Smearing and scattering limit the resolution of recordings of biological components, and, depending on the wavelength of the light used for illumination, the mapping component constitutes only a certain percentage of the changes in total light reflections.

## Acknowledgments

This work has been supported by the Wellcome Trust (050080/Z/97).

## References

[1] T. Bonhoeffer and A. Grinvald. Optical imaging based on intrinsic signals: The methodology. In A. Toga and J. C. Maziotta, editors, *Brain mapping: The methods*, pages 55–97, San Diego, CA, 1996. Academic Press, Inc.

[2] G. G. Blasdel and G. Salama. Voltage-sensitive dyes reveal a modular organization in monkey striate cortex. *Nature*, 321:579–585, 1986.

[3] G. G. Blasdel. Differential imaging of ocular dominance and orientation selectivity in monkey striate cortex. *J. Neurosci.*, 12:3115–3138, 1992.

[4] M. Stetter, T. Otto, T. Mueller, F. Sengpiel, M. Huebener, T. Bonhoeffer, and K. Obermayer. Temporal and spatial analysis of intrinsic signals from cat visual cortex. *Soc. Neurosci. Abstr.*, 23:455, 1997.

[5] I. Schießl, M. Stetter, J. E. W. Mayhew, S. Askew, N. McLoughlin, J. B. Levitt, J. S. Lund, and K. Obermayer. Blind separation of spatial signal patterns from optical imaging records. In J.-F. Cardoso, C. Jutten, and P. Loubaton, editors, *Proceedings of the ICA99 workshop*, volume 1, pages 179–184, 1999.

[6] A. J. Bell and T. J. Sejnowski. An information-maximization approach to blind separation and blind deconvolution. *Neural Computation*, 7:1129–1159, 1995.

[7] S. Amari. Neural learning in structured parameter spaces - natural riemannian gradient. In M. C. Mozer, M. I. Jordan, and T. Petsche, editors, *Advances in Neural Information Processing Systems*, volume 9, 1996.

[8] A. Hyvärinen and E. Oja. A fast fixed point algorithm for independent component analysis. *Neural Comput.*, 9:1483–1492, 1997.

[9] J. C. Platt and F. Faggin. Networks for the separation of sources that are superimposed and delayed. In J. E. Moody, S. J. Hanson, and R. P. Lippmann, editors, *Advances in Neural Information Processing Systems*, volume 4, pages 730–737, 1991.

[10] L. Molgedey and H. G. Schuster. Separation of a mixture of independent signals using time delayed correlations. *Phys. Rev. Lett.*, 72:3634–3637, 1994.

[11] S. M. Rüger. Stable dynamic parameter adaptation. In D. S. Touretzky, M. C. Mozer, and M. E. Hasselmo, editors, *Advances in Neural Information Processing Systems.*, volume 8, pages 225–231. MIT Press Cambridge, MA, 1996.

[12] K.-R. Müller, Philips P, and A. Ziehe. Jadetd: Combining higher-order statistics and temporal information for Blind Source Separation (with noise). In J.-F. Cardoso, C. Jutten, and P. Loubaton, editors, *Proceedings of the 1. ICA99 Workshop, Aussois*, volume 1, pages 87–92, 1999.

[13] B.-U. Koehler and R. Orglmeister. Independent component analysis using autoregressive models. In J.-F. Cardoso, C. Jutten, and P. Loubaton, editors, *Proceedings of the ICA99 workshop*, volume 1, pages 359–363, 1999.
